# Statistical Mechanics of the Mixture of Experts

**Kukjin Kang and Jong-Hoon Oh**
Department of Physics
Pohang University of Science and Technology
Hyoja San 31, Pohang, Kyongbuk 790-784, Korea
E-mail: `kkj,jhoh@galaxy.postech.ac.kr`

## Abstract

We study generalization capability of the mixture of experts learning from examples generated by another network with the same architecture. When the number of examples is smaller than a critical value, the network shows a symmetric phase where the role of the experts is not specialized. Upon crossing the critical point, the system undergoes a continuous phase transition to a symmetry breaking phase where the gating network partitions the input space effectively and each expert is assigned to an appropriate subspace. We also find that the mixture of experts with multiple level of hierarchy shows multiple phase transitions.

## 1 Introduction

Recently there has been considerable interest among neural network community in techniques that integrate the collective predictions of a set of networks[1, 2, 3, 4]. The mixture of experts [1, 2] is a well known example which implements the philosophy of divide-and-conquer elegantly. Whereas this model are gaining more popularity in various applications, there have been little efforts to evaluate generalization capability of these modular approaches theoretically. Here we present the first analytic study of generalization in the mixture of experts from the statistical

physics perspective. Use of statistical mechanics formulation have been focused on the study of feedforward neural network architectures close to the multilayer perceptron[5, 6], together with the VC theory[8]. We expect that the statistical mechanics approach can also be effectively used to evaluate more advanced architectures including mixture models.

In this letter we study generalization in the mixture of experts[1] and its variety with two-level hierarchy[2]. The network is trained by examples given by a teacher network with the same architecture. We find an interesting phase transition driven by symmetry breaking among the experts. This phase transition is closely related to the 'division-and-conquer' mechanism which this mixture model was originally designed to accomplish.

## 2   Statistical Mechanics Formulation for the Mixture of Experts

The mixture of experts[2] is a tree consisted of expert networks and gating networks which assign weights to the outputs of the experts. The expert networks sit at the leaves of the tree and the gating networks sit at its branching points of the tree. For the sake of simplicity, we consider a network with one gating network and two experts. Each expert produces its output $\mu_j$ as a generalized linear function of the $N$ dimensional input $\mathbf{x}$:

$$\mu_j = f(\mathbf{W}_j \cdot \mathbf{x}), \qquad j = 1, 2, \tag{1}$$

where $\mathbf{W}_j$ is a weight vector of the $j$ th expert with spherical constraint[5]. We consider a transfer function $f(x) = \mathrm{sgn}(x)$ which produces binary outputs. The principle of divide-and-conquer is implemented by assigning each expert to a subspace of the input space with different local rules. A gating network makes partitions in the input space and assigns each expert a weighting factor:

$$g_j(\mathbf{x}) = \Theta(\mathbf{V}_j \cdot \mathbf{x}), \tag{2}$$

where the gating function $\Theta(x)$ is the Heaviside step function. For two experts, this gating function defines a sharp boundary between the two subspace which is perpendicular to the vector $\mathbf{V}_1 = -\mathbf{V}_2 = \mathbf{V}$, whereas the softmax function used in the original literature [2] yield a soft boundary. Now the weighted output from the mixture of expert is written:

$$\mu(\mathbf{V}, \mathbf{W}; \mathbf{x}) = \sum_{j=1}^{2} g_j(\mathbf{x})\mu_j(\mathbf{x}). \tag{3}$$

The whole network as well as the individual experts generates binary outputs. Therefore, it can learn only dichotomy rules. The training examples are generated by a teacher with the same architecture as:

$$\sigma(\mathbf{x}_\mu) = \sum_{j=1}^{2} \Theta(\mathbf{V}_j^0 \cdot \mathbf{x})\mathrm{sgn}(\mathbf{W}_j^0 \cdot \mathbf{x}) , \tag{4}$$

where $V_j^0$ and $W_j^0$ are the weights of the $j$th gating network and the expert of the teacher.

The learning of the mixture of experts is usually interpreted probabilistically, hence the learning algorithm is considered as a maximum likelihood estimation. Learning algorithms originated from statistical methods such as the EM algorithm are often used. Here we consider Gibbs algorithm with noise level $T$ $(= 1/\beta)$ that leads to a Gibbs distribution of the weights after a long time:

$$P(\mathbf{V}, \mathbf{W}_j) = \frac{1}{Z} e^{-\beta E(\mathbf{V}, \mathbf{W}_j)}, \tag{5}$$

where $Z = \int d\mathbf{V} \, d\mathbf{W} \exp(-\beta E(\mathbf{V}, \mathbf{W}_j))$ is the partition function. Training both the experts and the gating network is necessary for a good generalization performance. The energy $E$ of the system is defined as a sum of errors over $P$ examples:

$$E(\mathbf{V}, \mathbf{W}_j) = \sum_{l=1}^{P} \epsilon(\mathbf{V}, \mathbf{W}_j; \mathbf{x}^l), \tag{6}$$

$$\epsilon(\mathbf{V}, \mathbf{W}_j; \mathbf{x}^l) = \Theta(-\mu(\mathbf{V}, \mathbf{W}_j; \mathbf{x}^l)\sigma(\mathbf{V}^0, \mathbf{W}_j^0; \mathbf{x}^l)). \tag{7}$$

The performance of the network is measured by the generalization function $\epsilon(\mathbf{V}, \mathbf{W}_j) = \int d\mathbf{x} \, \epsilon(\mathbf{V}, \mathbf{W}_j; \mathbf{x})$, where $\int d\mathbf{x}$ represents an average over the whole input space. The generalization error $\epsilon_g$ is defined by $\epsilon_g = \langle\!\langle\langle \epsilon(\mathbf{W}) \rangle_T \rangle\!\rangle$ where $\langle\!\langle \cdots \rangle\!\rangle$ denotes the quenched average over the examples and $\langle \cdots \rangle_T$ denotes the thermal average over the probability distribution of Eq. (5).

Since the replica calculation turns out to be intractable, we use the annealed approximation:

$$\langle\!\langle \log Z \rangle\!\rangle \simeq \log\langle\!\langle Z \rangle\!\rangle . \tag{8}$$

The annealed approximation is exact only in the high temperature limit, but it is known that the approximation usually gives qualitatively good results for the case of learning realizable rules[5, 6].

## 3   Generalization Curve and the Phase Transition

The generalization function $\epsilon(\mathbf{V}, \mathbf{W}_j)$ is can be written as a function of overlaps between the weight vectors of the teacher and the student:

$$\epsilon(\mathbf{V}, \mathbf{W}_j) = \sum_{i=1}^{2} \sum_{j=1}^{2} P_{ij} \epsilon_{ij} \tag{9}$$

where

$$P_{ij} = \frac{1}{2} \left( 1 - \frac{1}{\pi} \cos^{-1} R_{ij}^V \right) \tag{10}$$

$$\epsilon_{ij} = \frac{1}{\pi} \cos^{-1} R_{ij}, \tag{11}$$

and

$$R_{ij}^V = \frac{1}{N} \mathbf{V}_i \cdot \mathbf{V}_j^0, \qquad (12)$$

$$R_{ij} = \frac{1}{N} \mathbf{W}_i \cdot \mathbf{W}_j^0. \qquad (13)$$

is the overlap order parameters. Here, $P_{ij}$ is a probability that the $i$ th expert of the student learns from examples generated by the $j$ th expert of the teacher. It is a volume fraction in the input space where $\mathbf{V}_i \cdot \mathbf{x}$ and $\mathbf{V}_j^0 \cdot \mathbf{x}$ are both positive. For that particular examples, the $i$th expert of the student gives wrong answer with probability $\epsilon_{ij}$ with respect to the $j$ th expert of the teacher. We assume that the weight vectors of the teacher, $\mathbf{V}_0, \mathbf{W}_1^0$ and $\mathbf{W}_2^0$, are orthogonal to each other, then the overlap order parameters other than the ones shown above vanish. We use the symmetry properties of the network such as $R_V = R_{11}^V = R_{22}^V = -R_{12}^V$, $R = R_{11} = R_{22}$, and $r = R_{12} = R_{21}$.

The free energy also can be written as a function of three order parameters $R_V$, $R$, and $r$. Now we consider a thermodynamic limit where the dimension of the input space $N$ and the number of examples $P$ goes to infinity, keeping the ratio $\alpha = P/N$ finite. By minimizing the free energy with respect to the order parameters, we find the most probable values of the order parameters as well as the generalization error.

Fig 1.(a) plots the overlap order parameters $R_V$, $R$ and $r$ versus $\alpha$ at temperature $T = 5$. Examining the plot, we find an interesting phase transition driven by symmetry breaking among the experts. Below the phase transition point $\alpha_c = 51.5$, the overlap between the gating networks of the teacher and the student is zero ($R_V = 0$) and the overlaps between the experts are symmetric ($R = r$). In the symmetric phase, the gating network does not have enough examples to learn proper partitioning, so its performance is not much better than a random partitioning. Consequently each expert of the student can not specialize for the subspaces with a particular local rule given by an expert of the teacher. Each expert has to learn multiple linear rules with linear structure, which leads to a poor generalization performance. Unless more than a critical amount of examples is provided, the divide-and-conquer strategy does not work.

Upon crossing the critical point $\alpha_c$, the system undergoes a continuous phase transition to the symmetry breaking phase. The order parameter $R_V$, related to the goodness of partition, begins to increase abruptly and approaches 1 with increasing $\alpha$. The gating network now provides a better partition which is close to that of the teacher. The plot of order parameter $R$ and $r$, which is overlap between experts of teacher and student, branches at $\alpha_c$ and approaches 1 and 0 respectively. It means that each expert specializes its role by making appropriate pair with a particular expert of the teacher. Fig. 1(b) plots the generalization curve ($\epsilon_g$ versus $\alpha$) in the same scale. Though the generalization curve is continuous, the slope of the curve changes discontinuously at the transition point so that the generalization curve has

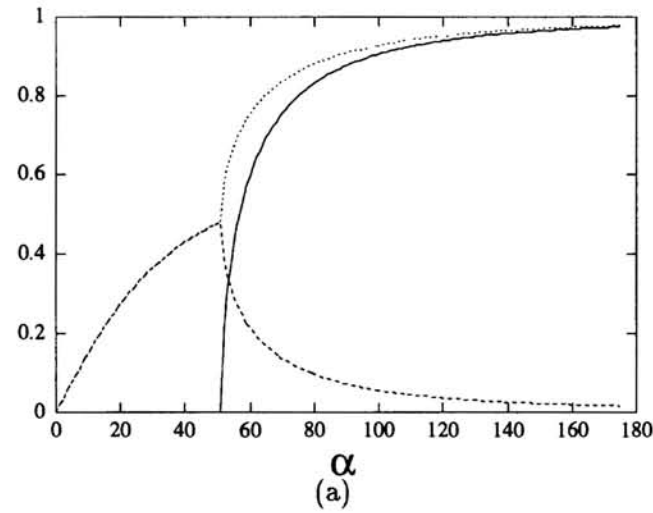

(a)

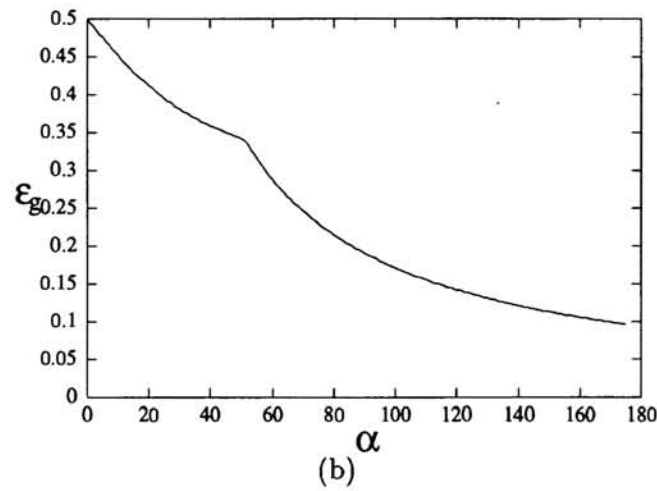

(b)

Figure 1: (a) The overlap order parameters $R_V$, $R$, $r$ versus $\alpha$ at $T = 5$. For $\alpha < \alpha_c = 51.5$, we find $R_V = 0$ (solid line that follows $x$ axis), and $R = r$ (dashed line). At the transition point, $R_V$ begins to increase abruptly, $R$ (dotted line) and $r$ (dashed line) branches, which approach 1 and 0 respectively . (b) The generalization curve ($\epsilon_g$ versus $\alpha$) for the mixture of experts in the same scale. A cusp at the transition point $\alpha_c$ is shown.

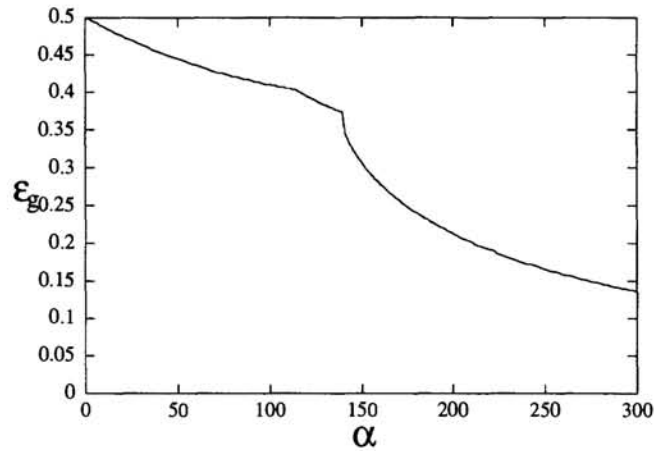

Figure 2: A typical generalization error curve for HME network with continuous weight. $T = 5$.

a cusp. The asymptotic behavior of $\epsilon_g$ at large $\alpha$ is given by:

$$\epsilon \simeq \frac{3}{1 - e^{-\beta}} \frac{1}{\alpha}, \tag{14}$$

where the $1/\alpha$ decay is often observed in learning of other feedforward networks.

## 4    The Mixture of Experts with Two-Level Hierarchy

We also study generalization in the hierarchical mixture of experts [2]. Consider a two-level hierarchical mixture of experts consisted of three gating networks and four experts. At the top level the tree is divided into two branch, and they are in turn divided into two branches at the lower level. The experts sit at the four leaves of the tree, and the three gating networks sit at the top and lower-level branching points. The network also learns from the training examples drawn from a teacher network with the same architecture.

FIG 2. (b) shows corresponding learning curve which has two cusps related to the phase transitions. For $\alpha < \alpha_{c1}$, the system is in the fully symmetric phase. The gating networks do not provide correct partition for the experts at both levels of hierarchy and the experts cannot specialize at all. All the overlaps with the weights of the teacher experts have the same value. The first phase transition at the smaller $\alpha_{c1}$ is related to the symmetry breaking by the top-level gating network. For $\alpha_{c1} < \alpha < \alpha_{c2}$, the top-level gating network partition the input space into two parts, but the lower-level gating network is not functioning properly. The overlap between the gating networks at the lower level of the tree and that of the teacher is still zero. The experts partially specialize into two groups. Specialization among the same group is not accomplished yet. The overlap order parameter $R_{ij}$ can

have two distinct values. The bigger one is the overlap with the two experts of the teacher for which the group is specializing, and the smaller is with the experts of the teacher which belong to the other group. At the second transition point $\alpha_{c2}$, the symmetry related to the lower-level hierarchy breaks. For $\alpha > \alpha_{c2}$, all the gating networks work properly and the input space is divided into four. Each expert makes appropriate pair with an expert of the teacher. Now the overlap order parameters can have three distinct values. The largest is the overlap with matching expert of teacher. The next largest is the overlap with the neighboring teacher expert in the tree hierarchy. The smallest is with the experts of the other group. The two phase transition result in the two cusps of the learning curve.

## 5 Conclusion

Whereas the phase transition of the mixture of experts can be interpreted as a symmetry breaking phenomenon which is similar to the one already observed in the committee machine and the multi-layer-perceptron[6, 7], the transition is novel in that it is continuous. This means that symmetry breaking is easier for the mixture of experts than in the multi-layer perceptron. This can be a big advantage in learning of highly nonlinear rules as we do not have to worry about the existence of local minima. We find that the hierarchical mixture of experts can have multiple phase transitions which are related to symmetry breaking at different levels. Note that symmetry breaking comes first from the higher-level branch, which is desirable property of the model.

We thank M. I. Jordan, L. K. Saul, H. Sompolinsky, H. S. Seung, H. Yoon and C. Kwon for useful discussions and comments. This work was partially supported by the Basic Science Special Program of the POSTECH Basic Science Research Institute.

## References

[1] R. A. Jacobs, M. I. Jordan, S. J. Nolwan, and G. E. Hinton, Neural Computation **3**, 79 (1991).

[2] M. I. Jordan, and R. A. Jacobs, Neural Computation **6**, 181 (1994).

[3] M.P. Perrone and L. N. Cooper, Neural Networks for Speech and Image Processing, R. J. Mammone. Ed., Chapman-Hill, London, 1993.

[4] D. Wolpert, Neural Networks, **5**, 241 (1992).

[5] H. S. Seung, H. Sompolinsky, and N. Tishby, Phys. Rev. A **45**, 6056 (1992).

[6] K. Kang, J.-H. Oh, C. Kwon and Y. Park, Phys. Rev. E **48**, 4805 (1993); K. Kang, J.-H. Oh, C. Kwon and Y. Park, Phys. Rev. E **54**, 1816 (1996).

[7] E. Baum and D. Haussler, Neural Computation **1**, 151 (1989).